# Rule Representations in a Connectionist Chunker

**David S. Touretzky      Gillette Elvgren III**
School of Computer Science
Carnegie Mellon University
Pittsburgh, PA 15213

## ABSTRACT

We present two connectionist architectures for chunking of symbolic rewrite rules. One uses backpropagation learning, the other competitive learning. Although they were developed for chunking the same sorts of rules, the two differ in their representational abilities and learning behaviors.

## 1  INTRODUCTION

Chunking is a process for generating, from a sequence of if-then rules, a more complex rule that accomplishes the same task in a single step. It has been used to explain incremental human performance improvement in a wide variety of cognitive, perceptual, and motor tasks (Newell, 1987). The SOAR production system (Laird, Newell, & Rosenbloom, 1987) is a classical AI computer program that implements a "unified theory of cognition" based on chunking.

SOAR's version of chunking is a symbolic process that examines the working memory trace of rules contributing to the chunk. In this paper we present two connectionist rule-following architectures that generate chunks a different way: they use incremental learning procedures to infer the environment in which the chunk should fire. The first connectionist architecture uses backpropagation learning, and has been described previously in (Touretzky, 1989a). The second architecture uses competitive learning. It exhibits more robust behavior than the previous one, at the cost of some limitations on the types of rules it can learn.

The knowledge to be chunked consists of context-sensitive rewrite rules on strings. For example, given the two rules

R1:     D → B / _ E    "change D to B when followed by E"
R2:     A → C / _ B    "change A to C when followed by B"

the model would go through the following derivation: ADE → (Rule R1) ABE → (Rule R2) CBE. Rule R1's firing is what enables rule R2 to fire. The model detects this and formulates a chunked rule (R1-R2) that can accomplish the same task in a single step:

R1-R2:     AD → CB / _ E

Once this chunk becomes active, the derivation will be handled in a single step, this way: ADE → (Chunk R1-R2) CBE. The chunk can also contribute to the formation of larger chunks.

## 2   CHUNKING VIA BACKPROPAGATION

Our first experiment, a three-layer backpropagation chunker, is shown in Figure 1. The input layer is a string buffer into which symbols are shifted one at a time, from the right. The output layer is a "change buffer" that describes changes to be made to the string. The changes supported are deletion of a segment, mutation of a segment, and insertion of a new segment. Combinations of these changes are also permitted.

Rules are implemented by hidden layer units that read the input buffer and write changes (via their $\alpha$ connections) into the change buffer. Then separate circuitry, not shown in the figure, applies the specified changes to the input string to update the state of the input buffer. The details of this string manipulation circuitry are given in (Touretzky, 1989b; Touretzky & Wheeler, 1990).

We will now go through the ADE derivation in detail. The model starts with an empty input buffer and two rules: R1 and R2.[1] After shifting the symbol A into the input buffer, no rule fires—the change buffer is all zeros. After shifting in the D, the input buffer contains AD, and again no rule fires. After shifting in the E the input buffer contains ADE, and rule R1 fires, writing a request in the change buffer to mutate input segment 2 (counting from the right edge of the buffer) to a B. The input buffer and change buffer states are saved in temporary buffers, and the string manipulation circuitry derives a new input buffer state, ABE. This now causes rule R2 to fire.[2] It writes a request into the change buffer to mutate segment 3 to a C. Since it was R1's firing that triggered R2, the conditions exist for chunk formation. The model combines R1's requested change with that of R2, placing the result in the "chunked change buffer" shown on the right in Figure 1. Backpropagation is used to teach the hidden layer that when it sees the input buffer pattern that triggered R1 (ADE in this case) it should produce via its $\beta$ connections the combined change pattern shown in the chunked change buffer.

The model's training is "self-supervised:" its own behavior (its history of rule firings) is the source of the chunks it acquires. It is therefore important that the chunking

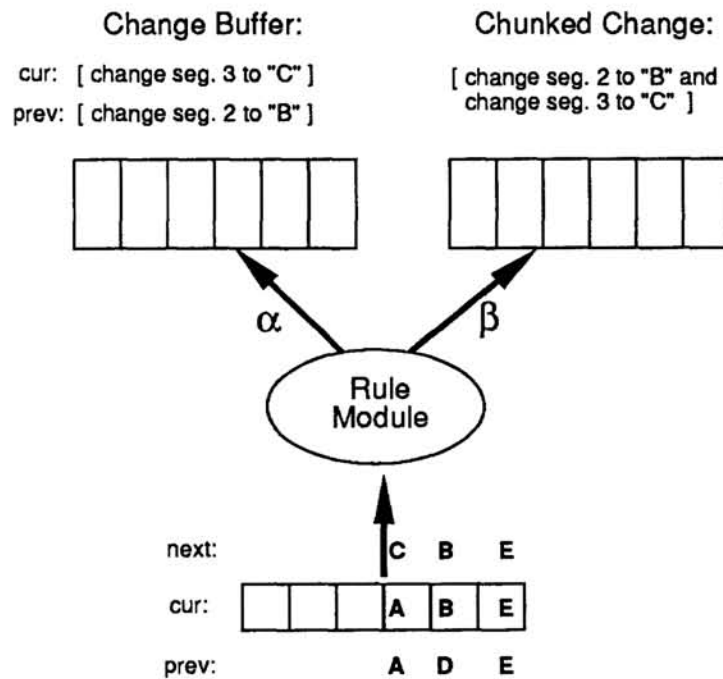

Change Buffer:
cur: [ change seg. 3 to "C" ]
prev: [ change seg. 2 to "B" ]

Chunked Change:
[ change seg. 2 to "B" and
change seg. 3 to "C" ]

α    β

Rule
Module

next:        C   B   E
cur:     A   B   E
prev:        A   D   E

**Figure 1:** Architecture of the backpropagation chunker.

process not introduce any behavioral errors during the intermediate stages of learning, since no external teacher is present to force the model back on track should its rule representations become corrupted. The original rules are represented in the $\alpha$ connections and the chunked rules are trained using the $\beta$ connections, but the two rule sets share the same hidden units and input connections, so interference can indeed occur. The model must actively preserve its $\alpha$ rules by continuous rehearsal: after each input presentation, backpropagation learning on a contrast-enhanced version of the $\alpha$ change pattern is used to counteract any interference caused by training on the $\beta$ patterns. Eventually, when the $\beta$ weights have been learned correctly, they can replace the $\alpha$ weights.

The parameters of the model were adjusted so that the initial rules had a distributed representation in the hidden layer, i.e., several units were responsible for implementing each rule. Analysis of the hidden layer representations after chunking revealed that the model had split off some of the R1 units to represent the R1-R2 chunk; the remainder were used to maintain the original R1 rule.

The primary flaw of this model is fragility. Constant rehearsal of the original rule set, and low learning rates, are required to prevent the $\alpha$ rules from being corrupted before the $\beta$ rules have been completely learned. Furthermore, it is difficult to form long rule chains, because each chunk further splits up the hidden unit population. Repeated splitting and retraining of hidden units proved difficult, but the model did manage to learn an R1-R2-R3 chunk that supersedes the R1-R2 chunk, so that ADE mutates directly to CFE. The third rule was:

R3:    $B \rightarrow F / C \_ E$    "change B to F when between C and E"

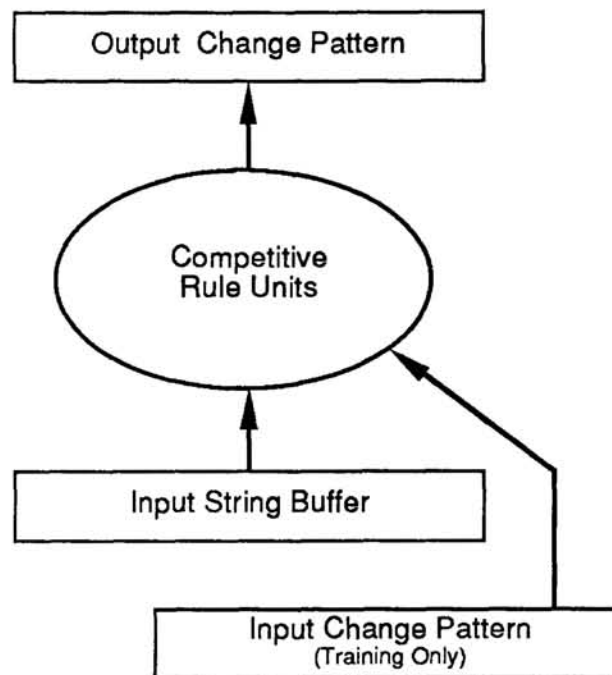

**Figure 2:** Architecture of the competitive learning chunker.

## 3   CHUNKING VIA COMPETITIVE LEARNING

Our second chunker, shown in Figure 2, minimizes interference between rules by using competitive learning to assign each rule a dedicated unit. As in the previous case, the model is taught its initial rules by showing it input buffer states and desired change buffer states. Chunks are then formed by running strings through the input buffer and watching for pairs of rules that fire sequentially. The model recruits new units for the chunks and teaches them to produce the new change buffer patterns (formed by composing the changes of the two original rules) in appropriate environments.

A number of technical problems had to be resolved in order to make this scheme work. First, we want to assign a separate unit to each rule, but not to each training example; otherwise the model will use too many units and not generalize well. Second, the encoding for letters we chose (see Table 1) is based on a Cartesian product, and so input patterns are highly overlapping and close together in Hamming space. This makes the job of the competitive learning algorithm more difficult. Third, there must be some way for chunks to take priority over the component rules from which they were formed, so that an input sequence like ADE fires the chunk R1-R2 rather than the original rule R1. As we trace through the operation of the chunker we will describe our solutions to these problems.

Rule units in the competitive layer are in one of three states: *inactive* (waiting to be recruited), *plastic* (currently undergoing learning), and *active* (weights finalized; ready to compete and fire.) They also contain a simple integrator (a counter) that is used to move them from the plastic to the active state. Initially all units are inactive and the counter

**Table 1:** Input code for both chunking models.

| | | | | | |
|---|---|---|---|---|---|
| A | 1 | 0 | 1 | 0 | 0 |
| B | 1 | 0 | 0 | 1 | 0 |
| C | 1 | 0 | 0 | 0 | 1 |
| D | 0 | 1 | 1 | 0 | 0 |
| E | 0 | 1 | 0 | 1 | 0 |
| F | 0 | 1 | 0 | 0 | 1 |

is zero. As in any competitive learning scheme, the rule units' input weights are kept normalized to unit vectors (Rumelhart & Zipser, 1986).

When the teacher presents a novel instance, we must determine if there is already some partially-trained rule unit whose weights should be shaped by this instance. Due to our choice of input code, it is not possible to reliably assign training instances to rule units based solely on the input pattern, because "similar" inputs (close in Hamming space) may invoke entirely different rules. Our solution is to use the desired change pattern as the primary index for selecting a pool of plastic rule units; the input buffer pattern is then used as a secondary cue to select the most strongly activated unit from this pool.

Let's consider what happens with the training example DE $\rightarrow$ BE. The desired change pattern "mutate segment 2 to a B" is fed to the competitive layer, and the network looks for plastic rule units whose change patterns exactly match the desired pattern.[3] If no such unit is found, one is allocated from the inactive pool, its status is changed to "plastic," its input buffer weights are set to match the pattern in the input buffer, and its change pattern input and and change pattern output weights are set according to the desired change pattern.

Otherwise, if a pool of suitable plastic units already exists, the input pattern DE is presented to the competitive layer and the selected plsatic units compete to see which most closely matches the input. The winning unit's input buffer weights are then adjusted by competitive learning to move the weight vector slightly closer to this input buffer vector. The unit's counter is also bumped.

Several presentations are normally required before a rule unit's input weights settle into their correct values, since the unit must determine from experience which input bit values are significant and which should be ignored. For example, rule S1 in Table 2 (the asterisk indicates a wildcard) can be learned from the training instances ACF and ADF, since as Table 1 shows, the letters C and D in the second segment have no bits in common. Therefore the learning algorithm will concentrate virtually all of the weight vector's magnitude in the connections that specify "A" as the first segment and "F" as the third.

Each time a rule unit's weights are adjusted by competitive learning, its counter is in-

cremented. When the counter reaches a suitable value (currently 25), the unit switches from the plastic to the active state. It is now ready to compete with other units for the right to fire; its weights will not change further.

We now consider the formation of the model's first chunk. Assume that rules R1 and R2 have been acquired successfully. The model is trained by running random strings through the input buffer and looking for sequences of rule firings. Suppose the model is presented with the input string BFDADE. R1 fires, producing BFDABE; this then causes R2 to fire, producing BFDCBE. The model proceeds to form a chunk. The combined change pattern specifies that the penultimate segment should be mutated to "B," and the antepenultimate to "C." Since no plastic rule unit's change pattern weights match this change, a fresh unit is allocated and its change buffer weights are set to reproduce this pattern. The unit's input weights are set to detect the pattern BFDADE.

After several more examples of the R1-R2 firing sequence, the competitive learning algorithm will discover that the first three input buffer positions can hold anything at all, but the last three always hold ADE. Hence the weight vector will be concentrated on the last three positions. When its counter reaches a value of 25, the rule unit will switch to the active state.

Now consider the next time an input ending in ADE is presented. The network is in performance mode now, so there is nothing in the input change buffer; the model is looking only at the input string buffer. The R1 unit will be fully satisfied by the input; its normalized weight vector concentrates on just the last two positions, "DE," which match exactly. The R1-R2 unit will also be fully satisfied; its normalized weight vector looks for the sequence ADE. The latter unit is the one we want to win the competition. We achieve this by scaling the activation function of competitive units by an additional factor: the degree of distributedness of the weight vector. Units that distribute their input weight over a larger number of connections likely represent complex chunks, and should therefore have their activation boosted over rules with narrowly focused input vectors.

Once the unit encoding the R1-R2 chunk enters the active state, its more distributed input weights assure that it will always win over the R1 unit for an input like ADE. The R1 unit may still be useful to keep around, though, to handle a case like FDE → FBE that does not trigger R2.

Sometimes a new chunk is learned that covers the same length input as the old, e.g., chunk R1-R2-R3 that maps ADE → CFE looks at exactly the same input positions as chunk R1-R2. We therefore introduce one additional term into the activation function. As part of the learning process, active units that contribute to the formation of a new chunk are given a permanent, very small inhibitory bias. This ensures that R1-R2 will always lose the competition to R1-R2-R3 once that chunk goes from plastic to active, even though their weights are distributed to an equal degree.

Another special case that needs to be handled is when the competitive algorithm wrongly splits a rule between two plastic units in the same pool, e.g., one unit might be assigned the cases {A,B,C}ADE, and the other the cases {D,E,F}ADE. (In other words, one unit looks for the bit pattern 10xxx in the first position, and the other unit looks for 01xxx.)

This is bad because it allows the weights of each unit to be more distributed than they need to be. To correct the problem, whenever a plastic unit wins a competition our algorithm makes sure that the nearest runner up is considerably less active than the winner. If its activation is too high, the runner up is killed. This causes the survivor to readjust its weights to describe the rule correctly, i.e., it will look for the input pattern ADE. If the runner up was killed incorrectly (meaning it is really needed for some other rule), it will be resurrected in response to future examples.

Finally, active units have a decay mechanism that is kept in check by the unit's firing occasionally. If a unit does not fire for a long time (200 input presentations), its weights decay to zero and it returns to the inactive state. This way, units representing chunks that have been superseded will eventually be recycled.

# 4  DISCUSSION

Each of the two learning architectures has unique advantages. The backpropagation learner can in principle learn arbitrarily complex rules, such as replacing a letter with its successor, or reversing a subset of the input string. Its use of a distributed rule representation allows knowledge of rule R1 to participate in the forming of the R1-R2 chunk. However, this representation is also subject to interference effects, and as is often the case with backprop, learning is slow.

The competitive architecture learns very quickly. It can form a greater number of chunks, and can handle longer rule chains, since it avoids inteference by assigning a dedicated unit to each new rule it learns.

Both learners are sensitive to changes in the distribution of input strings; new chunks can form any time they are needed. Chunks that are no longer useful in the backprop model will eventually fade away due to non-rehearsal; the hidden units that implement these chunks will be recruited for other tasks. The competitive chunker uses a separate decay mechanism to recycle chunks that have been superseded.

This work shows that connectionist techniques can yield novel and interesting solutions to symbol processing problems. Our models are based on a sequence manipulation architecture that uses a symbolic description of the changes to be made (via the change buffer), but the precise environments in which rules apply are never explicitly represented. Instead they are induced by the learning algorithm from examples of the models' own behavior. Such self-supervised learning may play an important role in cognitive development. Our work shows that it is possible to correctly chunk knowledge even when one cannot predict the precise environment in which the chunks should apply.

### Acknowledgements

This research was supported by a contract from Hughes Research Laboratories, by the Office of Naval Research under contract number N00014-86-K-0678, and by National Science Foundation grant EET-8716324. We thank Allen Newell, Deirdre Wheeler, and Akihiro Hirai for helpful discussions.

**Table 2:** Initial rule set for the competitive learning chunker.

```
S1:   A*F        → B*F
S2:   BD         → BF
S3:   {D,E,F}*E  → {A,B,C}*A
S4:   {B,E}B     → CB
S5:   {A,D}C     → {C,F}C
```

**Table 3:** Chunks formed by the competitive learning chunker.

| Chunk | (Component Rules) |
|---|---|
| EA*F → CB*F | (S1,S4) |
| ABD → CBF | (S1,S2,S4) |
| AADF → CBFF | (S1,S2,S1,S4) |
| BE*E → CB*A | (S3,S4) |
| DEB → FEB | (S4,S5) |

## Footnotes

[1] The initial rule set is installed by an external teacher using backpropagation.

[2] Note that R1 applies to positions 1 and 2 of the buffer (counting from the right edge), while R2 applies to positions 2 and 3. Rules are represented in a position-independent manner, allowing them to apply anywhere in the buffer that their environment is satisfied. The mechanism for achieving this is explained in (Touretzky, 1989a).

[3]The units' thresholds are raised so that they can only become active if their weight vectors match the input change buffer vector exactly.

### References

Laird, J. E., Newell, A., and Rosenbloom, P. S. (1987) Soar: An architecture for general intelligence. *Artificial Intelligence* 33(1):1-64.

Newell, A. (1987) The 1987 William James Lectures: Unified Theories of Cognition. Given at Harvard University.

Rumelhart, D E., and Zipser, D. (1986) Feature discovery by competitive learning. In D. E. Rumelhart and J. L. McClelland (eds.), *Parallel Distributed Processing: Explorations in the Microstructure of Cognition*. Cambridge, MA: MIT Press.

Touretzky, D. S. (1989a) Chunking in a connectionist network. *Proceedings of the Eleventh Annual Conference of the Cognitive Science Society*, pp. 1-8. Hillsdale, NJ: Erlbaum.

Touretzky, D. S. (1989b) Towards a connectionist phonology: the "many maps" approach to sequence manipulation. *Proceedings of the Eleventh Annual Conference of the Cognitive Science Society*, pp. 188-195. Hillsdale, NJ: Erlbaum.

Touretzky, D. S., and Wheeler, D. W. (1990) A computational basis for phonology. In D. S. Touretzky (ed.), *Advances in Neural Information Processing Systems 2*. San Mateo, CA: Morgan Kaufmann.
